# Escaping the Convex Hull with Extrapolated Vector Machines.

**Patrick Haffner**

AT&T Labs-Research, 200 Laurel Ave, Middletown, NJ 07748

`haffner@research.att.com`

## Abstract

Maximum margin classifiers such as Support Vector Machines (SVMs) critically depends upon the convex hulls of the training samples of each class, as they implicitly search for the minimum distance between the convex hulls. We propose Extrapolated Vector Machines (XVMs) which rely on extrapolations outside these convex hulls. XVMs improve SVM generalization very significantly on the MNIST [7] OCR data. They share similarities with the Fisher discriminant: maximize the inter-class margin while minimizing the intra-class disparity.

## 1 Introduction

Both intuition and theory [9] seem to support that the best linear separation between two classes is the one that maximizes the margin. But is this always true? In the example shown in Fig.(1), the maximum margin hyperplane is $W_0$; however, most observers would say that the separating hyperplane $W_1$ has better chances to generalize, as it takes into account the expected location of additional training sam-

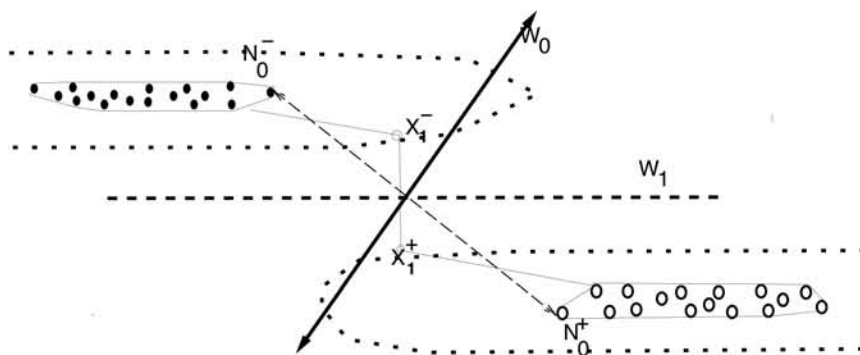

Figure 1: Example of separation where the large margin is undesirable. The convex hull and the separation that corresponds to the standard SVM use plain lines while the extrapolated convex hulls and XVMs use dotted lines.

ples. Traditionally, to take this into account, one would estimate the distribution of the data. In this paper, we just use a very elementary form of extrapolation ("the poor man variance") and show that it can be implemented into a new extension to SVMs that we call Extrapolated Vector Machines (XVMs).

## 2  Adding Extrapolation to Maximum Margin Constraints

This section states extrapolation as a constrained optimization problem and computes a simpler dual form.

Take two classes $\mathcal{C}_+$ and $\mathcal{C}_-$ with $y_+ = +1$ and $y_- = -1$ [1] as respective targets. The $N$ training samples $\{(\mathbf{x}_i, y_i); 1 \leq i \leq N\}$ are separated with a margin $\rho$ if there exists a set of weights $\mathbf{w}$ such that $\|\mathbf{w}\| = 1$ and

$$\forall k \in \{+, -\}, \ \forall i \in \mathcal{C}_k, \ y_k(\mathbf{w}\cdot\mathbf{x}_i + b) \geq \rho \tag{1}$$

SVMs offer techniques to find the weights $\mathbf{w}$ which maximize the margin $\rho$. Now, instead of imposing the margin constraint on each training point, suppose that for two points in the same class $\mathcal{C}_k$, we require any possible extrapolation within a range factor $\eta_k \geq 0$ to be larger than the margin:

$$\forall i, j \in \mathcal{C}_k, \ \forall \lambda \in [-\eta_k, 1+\eta_k], \ y_k\left(\mathbf{w}\cdot(\lambda\mathbf{x}_i + (1-\lambda)\mathbf{x}_j) + b\right) \geq \rho \tag{2}$$

It is sufficient to enforce the constraints at the end of the extrapolation segments, and

$$\forall i, j \in \mathcal{C}_k, \ y_k\left(\mathbf{w}\cdot((\eta_k+1)\mathbf{x}_i - \eta_k\mathbf{x}_j) + b\right) \geq \rho \tag{3}$$

Keeping the constraint over each pair of points would result in $N^2$ Lagrange multipliers. But we can reduce it to a double constraint applied to each single point. If follows from Eq.(3) that:

$$(\eta_+ + 1)\min_{i\in\mathcal{C}_+}\left((\mathbf{w}\cdot\mathbf{x}_i)\right) + b = \rho + \eta_+\max_{j\in\mathcal{C}_+}\left((\mathbf{w}\cdot\mathbf{x}_j)\right) \tag{4}$$

$$(\eta_- + 1)\min_{i\in\mathcal{C}_-}\left(-(\mathbf{w}\cdot\mathbf{x}_i)\right) - b = \rho + \eta_-\max_{j\in\mathcal{C}_-}\left(-(\mathbf{w}\cdot\mathbf{x}_j)\right) \tag{5}$$

We consider $\mu_k = \max_{j\in\mathcal{C}_k}(y_k(\mathbf{w}\cdot\mathbf{x}_j))$ and $\nu_k = \min_{j\in\mathcal{C}_k}(y_k(\mathbf{w}\cdot\mathbf{x}_j))$ as optimization variables. By adding Eq.(4) and (5), the margin becomes

$$2\rho = \sum_k\left((\eta_k+1)\nu_k - \eta_k\mu_k\right) = \sum_k\left(\nu_k - \eta_k(\mu_k - \nu_k)\right) \tag{6}$$

Our problem is to maximize the margin under the double constraint:

$$\forall i \in \mathcal{C}_k, \ \nu_k \leq y_k(\mathbf{w}\cdot\mathbf{x}_i) \leq \mu_k$$

In other words, the extrapolated margin maximization is equivalent to squeezing the points belonging to a given class between *two* hyperplanes. Eq.(6) shows that $\rho$ is maximized when $\nu_k$ is maximized while $\mu_k - \nu_k$ is minimized.

Maximizing the margin over $\mu_k$, $\nu_k$ and $\mathbf{w}$ with Lagrangian techniques gives us the following *dual problem*:

$$\min_{\beta,\hat{\beta}}\left\|\sum_k y_k\left((\eta_k+1)\sum_{i\in\mathcal{C}_k}\beta_i\mathbf{x}_i - \eta_k\sum_{i\in\mathcal{C}_k}\hat{\beta}_i\mathbf{x}_i\right)\right\|^2 \quad \begin{cases} 0 \leq \beta_i, \ \sum_{i\in\mathcal{C}_k}\beta_i = 1 \\ 0 \leq \hat{\beta}_i, \ \sum_{i\in\mathcal{C}_k}\hat{\beta}_i = 1 \end{cases} \tag{7}$$

Compared to the standard SVM formulation, we have two sets of support vectors. Moreover, the Lagrange multipliers that we chose are normalized differently from the traditional SVM multipliers (note that this is one possible choice of notation, see Section.6 for an alternative choice). They sum to 1 and allow and interesting geometric interpretation developed in the next section.

## 3 Geometric Interpretation and Iterative Algorithm

For each class $k$, we define the nearest point to the other class convex hull along the direction of $\mathbf{w}$: $\mathbf{N}_k = \sum_{i \in \mathcal{C}_k} \beta_i \mathbf{x}_i$. $\mathbf{N}_k$ is a combination of the *internal* support vectors that belong to class $k$ with $\beta_i > 0$. At the minimum of (7), because they correspond to non zero Lagrange multipliers, they fall on the internal margin $y_k(\mathbf{w} \cdot \mathbf{x}_i) = \nu_k$; therefore, we obtain $\nu_k = y_k \mathbf{w} \cdot \mathbf{N}_k$.

Similarly, we define the furthest point $\mathbf{F}_k = \sum_{i \in \mathcal{C}_k} \hat{\beta}_i \mathbf{x}_i$. $\mathbf{F}_k$ is a combination of the *external* support vectors, and we have $\mu_k = y_k \mathbf{w} \cdot \mathbf{F}_k$.

The dual problem is equivalent to the distance minimization problem

$$\min_{\mathbf{N}_k, \mathbf{F}_k \in \mathcal{H}^k} \left\| \sum_k y_k \left( (\eta_k + 1)\mathbf{N}_k - \eta_k \mathbf{F}_k \right) \right\|^2$$

where $\mathcal{H}^k$ is the convex hull containing the examples of class $k$.

It is possible to solve this optimization problem using an iterative Extrapolated Convex Hull Distance Minimization (XCHDM) algorithm. It is an extension of the Nearest Point [5] or Maximal Margin Perceptron [6] algorithms. An interesting geometric interpretation is also offered in [3]. All the aforementioned algorithms search for the points in the convex hulls of each class that are the nearest to each other ($N_0^+$ and $N_0^-$ on Fig.1), the maximal margin weight vector $\mathbf{w} = N_0^+ - N_0^-$.

XCHDM look for nearest points in the *extrapolated* convex hulls ($\mathbf{X}^+{}_1$ and $\mathbf{X}^-{}_1$ on Fig.1). The extrapolated nearest points are $\mathbf{X}_k = \eta_k \mathbf{N}_k - \eta_k \mathbf{F}_k$. Note that they can be outside the convex hull because we allow negative contribution from external support vectors. Here again, the weight vector can be expressed as a difference between two points $\mathbf{w} = \mathbf{X}^+ - \mathbf{X}^-$. When the data is non-separable, the solution is trivial with $\mathbf{w} = \mathbf{0}$. With the double set of Lagrange multipliers, the description of the XCHDM algorithm is beyond the scope of this paper. XCHDM with $\eta_k = 0$ are simple SVMs trained by the same algorithm as in [6].

An interesting way to follow the convergence of the XCHDM algorithm is the following. Define the extrapolated *primal margin*

$$\gamma_1^* = 2\rho = \sum_k \left( (\eta_k + 1)\nu_k - \eta_k \mu_k \right)$$

and the *dual margin*

$$\gamma_2^* = \left\| \mathbf{X}^+ - \mathbf{X}^- \right\|$$

Convergence consists in reducing the *duality gap* $\gamma_2^* - \gamma_1^*$ down to zero. In the rest of the paper, we will measure convergence with the *duality ratio* $r = \frac{\gamma_1^*}{\gamma_2^*}$.

To determine the threshold to compute the classifier output class $\mathrm{sign}(\mathbf{w} \cdot \mathbf{x} + b)$ leaves us with two choices. We can require the separation to happen at the center of the primal margin, with the *primal threshold* (subtract Eq.(5) from Eq.(4))

$$b_1 = -\frac{1}{2} \sum_k y_k \left( (\eta_k + 1)\nu_k - \eta_k \mu_k \right)$$

or at the center of the dual margin, with the *dual threshold*

$$b_2 = -\frac{1}{2}\mathbf{w} \cdot \sum_k \left( (\eta_k+1)\mathbf{N}_k - \eta_k \mathbf{F}_k \right) = -\frac{1}{2}\left( \left\| \mathbf{X}^+ \right\|^2 - \left\| \mathbf{X}^- \right\|^2 \right)$$

Again, at the minimum, it is easy to verify that $b_1 = b_2$. When we did not let the XCHDM algorithm converge to the minimum, we found that $b_1$ gave better generalization results.

Our standard stopping heuristic is numerical: stop when the duality ratio gets over a fixed value (typically between 0.5 and 0.9).

The only other stopping heuristic we have tried so far is based on the following idea. Define the set of extrapolated pairs as $\{(\eta_k+1)\mathbf{x}_i - \eta_k\mathbf{x}_j; 1 \le i, j \le N\}$. Convergence means that we find extrapolated support pairs that contain every extrapolated pair on the correct side of the margin. We can relax this constraint and stop when the *extrapolated* support pairs contain every vector. This means that $\gamma_2^*$ must be lower than the primal *true* margin along $\mathbf{w}$ (measured on the non-extrapolated data) $\gamma_1 = \nu^+ + \nu^-$. This causes the XCHDM algorithm to stop long before $\gamma_2^*$ reaches $\gamma_1^*$ and is called the *hybrid* stopping heuristic.

## 4 Beyond SVMs and discriminant approaches.

Kernel Machines consist of any classifier of the type $f(\mathbf{x}) = \sum_i y_i \alpha_i K(\mathbf{x}, \mathbf{x}_i)$. SVMs offer one solution among many others, with the constraint $\alpha_i > 0$.

XVMs look for solutions that no longer bear this constraint. While the algorithm described in Section 2 converges toward a solution where vectors act as support of margins (internal and external), experiments show that the performance of XVMs can be significantly improved if we stopped before full convergence. In this case, the vectors with $\alpha_i \ne 0$ do not line up onto any type of margin, and should not be called *support* vectors.

The extrapolated margin contains terms which are caused by the extrapolation and are proportional to the width of each class along the direction of $\mathbf{w}$. We would observe the same phenomenon if we had trained the classifier using Maximum Likelihood Estimation (MLE) (replace class width with variance). In both MLE and XVMs, examples which are the furthest from the decision surface play an important role. XVMs suggest an explanation why.

Note also that like the Fisher discriminant, XVMs look for the projection that maximizes the inter-class variance while minimizing the intra-class variances.

## 5 Experiments on MNIST

The MNIST OCR database contains 60,000 handwritten digits for training and 10,000 for testing (the testing data can be extended to 60,000 but we prefer to keep unseen test data for final testing and comparisons). This database has been extensively studied on a large variety of learning approaches [7]. It lead to the first SVM "success story"[2], and results have been improved since then by using knowledge about the invariance of the data [4].

The input vector is a list of 28x28 pixels ranging from 0 to 255. Before computing the kernels, the input vectors are normalized to 1: $\bar{\mathbf{x}} = \frac{\mathbf{x}}{\|\mathbf{x}\|}$.

Good polynomial kernels are easy to define as $\bar{K}_p(\mathbf{x}, \mathbf{y}) = (\bar{\mathbf{x}} \cdot \bar{\mathbf{y}})^p$. We found these normalized kernels to outperform the unnormalized kernels $K_p(\mathbf{x}, \mathbf{y}) = (a(\mathbf{x} \cdot \mathbf{y}) + b)^p$

that have been traditionally used for the MNIST data significantly. For instance, the baseline error rate with $\bar{K}_4$ is below 1.2%, whereas it hovers around 1.5% for $K_4$ (after choosing optimal values for $a$ and $b$)[2].

We also define normalized Gaussian kernels:

$$\tilde{K}_p(\mathbf{x}, \mathbf{y}) = \exp\left(-\frac{p}{2}\|\bar{\mathbf{x}} - \bar{\mathbf{y}}\|^2\right) = \left[\exp\left(\bar{\mathbf{x}}\cdot\bar{\mathbf{y}}-1\right)\right]^p. \tag{8}$$

Eq.(8) shows how they relate to normalized polynomial kernels: when $\bar{\mathbf{x}}\cdot\bar{\mathbf{y}} \ll 1$, $\tilde{K}_p$ and $\bar{K}_p$ have the same asymptotic behavior. We observed that on MNIST, the performance with $\tilde{K}_p$ is very similar to what is obtained with unnormalized Gaussian kernels $K_\sigma(\mathbf{x}, \mathbf{y}) = \exp-(\frac{\mathbf{x}-\mathbf{y}}{\sigma})^2$. However, they are easier to analyze and compare to polynomial kernels.

MNIST contains 1 class per digit, so the total number of classes is M=10. To combine binary classifiers to perform multiclass classifications, the two most common approaches were considered.

- In the *one-vs-others* case (1vsR), we have one classifier per class $c$, with the positive examples taken from class $c$ and negative examples form the other classes. Class $c$ is recognized when the corresponding classifier yields the largest output.

- In the *one-vs-one* case (1vs1), each classifier only discriminates one class from another: we need a total of $\frac{(M(M-1))}{2} = 45$ classifiers.

Despite the effort we spent on optimizing the recombination of the classifiers [8] [3], 1vsR SVMs (Table 1) perform significantly better than 1vs1 SVMs (Table 2). [4]

For each trial, the number of errors over the 10,000 test samples (#err) and the total number of support vectors(#SV) are reported. As we only count SVs which are shared by different classes once, this predicts the test time. For instance, 12,000 support vectors mean that 20% of the 60,000 vectors are used as support.

Preliminary experiments to choose the value of $\eta_k$ with the hybrid criterion show that the results for $\eta_k = 1$ are better than $\eta_k = 1.5$ in a statistically significant way, and slightly better than $\eta_k = 0.5$. We did not consider configurations where $\eta^+ \neq \eta^-$; however, this would make sense for the assymetrical 1vsR classifiers.

XVM gain in performance over SVMs for a given configuration ranges from 15% (1vsR in Table 3) to 25% (1vs1 in Table 2).

$$\frac{1}{\rho^2} = \sum_{c \leq M} \frac{1}{\rho_c^2}$$

The fact that this margin predicts generalization is "justified" by Theorem 1 in [8].

| Kernel | Duality Ratio stop | | | | | |
| | 0.40 | | 0.75 | | 0.99 | |
| | #err | #SV | #err | #SV | #err | #SV |
|---|---|---|---|---|---|---|
| $\bar{K}_3$ | 136 | 8367 | 136 | 11132 | 132 | 13762 |
| $\bar{K}_4$ | 127 | 8331 | 117 | **11807** | 119 | 15746 |
| $\bar{K}_5$ | 125 | 8834 | 119 | 12786 | 119 | 17868 |
| $\bar{K}_9$ | 136 | 13002 | 137 | 18784 | 141 | 25953 |
| $\tilde{K}_2$ | 147 | 9014 | 128 | 11663 | 131 | 13918 |
| $\tilde{K}_4$ | 125 | 8668 | 119 | 12222 | 117 | 16604 |
| $\tilde{K}_5$ | 125 | 8944 | 125 | 12852 | 125 | 18085 |

Table 1: SVMs on MNIST with 10 1vsR classifiers

| Kernel | SVM/ratio at 0.99 | | XVM/Hybrid | |
| | #err | #SV | #err | #SV |
|---|---|---|---|---|
| $\bar{K}_3$ | 138 | 11952 | 117 | 17020 |
| $\bar{K}_4$ | 135 | 13526 | 110 | 16066 |
| $\bar{K}_5$ | 191 | 13526 | 114 | 15775 |

Table 2: SVM/XVM on MNIST with 45 1vs1 classifiers

The 103 errors obtained with $\bar{K}_4$ and $r = 0.5$ in Table 3 represent only about 1% error: this is the *lowest error ever reported for any learning technique without a priori knowledge* about the fact that the input data corresponds to a pixel map (the lowest reproducible error previously reported was 1.2% with SVMs and polynomials of degree 9 [4], it could be reduced to 0.6% by using invariance properties of the pixel map). The downside is that XVMs require 4 times as many support vectors as standards SVMs.

Table 3 compares stopping according to the duality ratio and the hybrid criterion. With the duality ratio, the best performance is most often reached with $r = 0.50$ (if this happens to be consistently true, validation data to decide when to stop could be spared). The hybrid criterion does not require validation data and yields errors that, while higher than the best XVM, are lower than SVMs and only require a few more support vectors. It takes fewer iterations to train than SVMs. One way to interpret this hybrid stopping criterion is that we stop when interpolation in some (but not all) directions account for all *non-interpolated* vectors. This suggest that interpolation is only desirable in a few directions.

XVM gain is stronger in the 1vs1 case (Table 2). This suggests that extrapolating on a convex hull that contains several different classes (in the 1vsR case) may be undesirable.

| Kernel | Duality Ratio stop | | | | | | Hybrid. Stop Crit. | |
| | 0.40 | | 0.50 | | 0.75 | | | |
| | #err | #SV | #err | #SV | #err | #SV | #err | #SV |
|---|---|---|---|---|---|---|---|---|
| $\bar{K}_3$ | 118 | 46662 | 111 | 43819 | 116 | 50216 | 125 | 20604 |
| $\bar{K}_4$ | 112 | 40274 | **103** | 43132 | 110 | 52861 | 107 | 18002 |
| $\bar{K}_5$ | 109 | 36912 | 106 | 44226 | 110 | 49383 | 107 | 17322 |
| $\bar{K}_9$ | 128 | 35809 | 126 | 39462 | 131 | 50233 | 125 | 19218 |
| $\tilde{K}_2$ | 114 | 43909 | 114 | 46905 | 114 | 53676 | 119 | 20152 |
| $\tilde{K}_4$ | 108 | 36980 | 111 | 40329 | 114 | 51088 | 108 | *16895* |

Table 3: XVMs on MNIST with 10 1vsR classifiers

## 6 The Soft Margin Case

MNIST is characterized by the quasi-absence of outliers, so to assume that the data is fully separable does not impair performance at all. To extend XVMs to non-separable data, we first considered the traditional approaches of adding slack variables to allow margin constraints to be violated. The most commonly used approach with SVMs adds linear slack variables to the unitary margin. Its application to the XVM requires to give up the weight normalization constraint, so that the usual unitary margin can be used in the constraints [9].

Compared to standard SVMs, a new issue to tackle is the fact that each constraint corresponds to a pair of vectors: ideally, we should handle $N^2$ slack variables $\xi_{ij}$. To have linear constraints that can be solved with KKT, we need to have the decomposition $\xi_{ij} = (\eta_k+1)\xi_i + \eta_k\xi_j^*$ (factors $(\eta_k+1)$ and $\eta_k$ are added here to ease later simplifications).

Similarly to Eq.(3), the constraint on the extrapolation from any pair of points is

$$\forall i,j \in \mathcal{C}_k, \; y_k\left(\mathbf{w}\cdot\left((\eta_k+1)\mathbf{x}_i - \eta_k\mathbf{x}_j\right)+b\right) \geq 1 - (\eta_k+1)\xi_i - \eta_k\xi_j^* \text{ with } \xi_i, \xi_j^* \geq 0 \quad (9)$$

Introducing $\mu_k = \max\limits_{j \in \mathcal{C}_k}\left(y_k(\mathbf{w}\cdot\mathbf{x}_j+b) - \xi_j^*\right)$ and $\nu_k = \min\limits_{i \in \mathcal{C}_k}\left(y_k(\mathbf{w}\cdot\mathbf{x}_i+b) + \xi_i\right)$, we obtain the simpler double constraint

$$\forall i \in \mathcal{C}_k, \; \nu_k - \xi_i \leq y_k(\mathbf{w}\cdot\mathbf{x}_i+b) \leq \mu_k + \xi_i^* \text{ with } \xi_i, \xi_i^* \geq 0 \quad (10)$$

It follows from Eq.(9) that $\mu_k$ and $\nu_k$ are tied through $(1+\eta_k)\nu_k = 1+\eta_k\mu_k$

If we fix $\mu_k$ (and thus $\nu_k$) instead of treating it as an optimization variable, it would amount to a standard SVM regression problem with $\{-1,+1\}$ outputs, the width of the asymmetric $\epsilon$-insensitive tube being $\mu_k - \nu_k = \frac{\mu_k-1}{(\eta_k+1)}$.

This remark makes it possible for the reader to verify the results we reported on MNIST. Using the publicly available SVM software SVM*torch* [1] with $C = 10$ and $\epsilon = 0.1$ as the width of the $\epsilon$-tube yields a 10-class error rate of 1.15% while the best performance using SVM*torch* in classification mode is 1.3% (in both cases, we use Gaussian kernels with parameter $\sigma = 1650$).

An explicit minimization on $\mu_k$ requires to add to the standard SVM regression problem the following constraint over the Lagrange multipliers (we use the same notation as in [9]):

$$\sum_{y_i=1} \alpha_i = \sum_{y_i=-1} \alpha_i \text{ and } \sum_{y_i=1} \alpha_i^* = \sum_{y_i=-1} \alpha_i^*$$

Note that we still have the standard regression constraint $\sum \alpha_i = \sum \alpha_i^*$

This has not been implemented yet, as we question the pertinence of the $\xi_i^*$ slack variables for XVMs. Experiments with SVM*torch* on a variety of tasks where non-zero slacks are required to achieve optimal performance (Reuters, UCI/Forest, UCI/Breast cancer) have not shown significant improvement using the regression mode while we vary the width of the $\epsilon$-tube.

Many experiments on SVMs have reported that removing the outliers often gives efficient and sparse solutions. The early stopping heuristics that we have presented for XVMs suggest strategies to avoid learning (or to unlearn) the outliers, and this is the approach we are currently exploring.

# 7 Concluding Remarks

This paper shows that large margin classification on extrapolated data is equivalent to the addition of the minimization of a second *external* margin to the standard SVM approach. The associated optimization problem is solved efficiently with convex hull distance minimization algorithms. A 1% error rate is obtained on the MNIST dataset: it is the lowest ever obtained without a-priori knowledge about the data.

We are currently trying to identify what other types of dataset show similar gains over SVMs, to determine how dependent XVM performance is on the facts that the data is separable or has invariance properties. We have only explored a few among the many variations the XVM models and algorithms allow, and a justification of why and when they generalize would help model selection. Geometry-based algorithms that handle potential outliers are also under investigation.

Learning Theory bounds that would be a function of both the margin and some form of variance of the data would be necessary to predict XVM generalization and allow us to also consider the extrapolation factor $\eta$ as an optimization variable.

## Footnotes

[1]In this paper, it is necessary to index the outputs $y$ with the class $k$ rather than the more traditional sample index $i$, as extrapolation constraints require two examples to belong to the same class. The resulting equations are more concise, but harder to read.

[2]This may partly explain a nagging mystery among researchers working on MNIST: how did Cortes and Vapnik [2] obtain 1.1% error with a degree 4 polynomial ?

[3]We compared the *Max Wins* voting algorithm with the *DAGSVM* decision tree algorithm and found them to perform equally, and worse than 1vsR SVMs. This is is surprising in the light of results published on other tasks [8], and would require further investigations beyond the scope of this paper.

[4]Slightly better performance was obtained with a new algorithm that uses the incremental properties of our training procedure (this is be the performance reported in the tables). In a transductive inference framework, treat the test example as a training example: for each of the M possible labels, retrain the M among $\frac{(M(M-1))}{2}$ classifiers that use examples with such label. The best label will be the one that causes the smallest increase in the multiclass margin $\rho$ such that it combines the classifier margins $\rho_c$ in the following manner

# References

[1] R. Collobert and S. Bengio. Support vector machines for large-scale regression problems. Technical Report IDIAP-RR-00-17, IDIAP, 2000.

[2] C. Cortes and V. Vapnik. Support vector networks. *Machine Learning*, 20:1–25, 1995.

[3] D. Crisp and C.J.C. Burges. A geometric interpretation of $\nu$-SVM classifiers. In *Advances in Neural Information Processing Systems 12, S. A. Solla, T. K. Leen, K.-R. Mller, eds*, Cambridge, MA, 2000. MIT Press.

[4] D. DeCoste and B. Schoelkopf. Training invariant support vector machines. *Machine Learning, special issue on Support Vector Machines and Methods*, 2001.

[5] S.S. Keerthi, S.K. Shevade, C. Bhattacharyya, and K.R.K. Murthy. A fast iterative nearest point algorithm for support vector machine classifier design. *IEEE transactions on neural networks*, 11(1):124 –136, jan 2000.

[6] A. Kowalczyk. Maximal margin perceptron. In *Advances in Large Margin Classifiers, Smola, Bartlett, Schlkopf, and Schuurmans, editors*, Cambridge, MA, 2000. MIT Press.

[7] Y. LeCun, L. Bottou, Y. Bengio, and P. Haffner. Gradient-based learning applied to document recognition. *proceedings of the IEEE*, 86(11), 1998.

[8] J. Platt, N. Christianini, and J. Shawe-Taylor. Large margin dags for multiclass classification. In *Advances in Neural Information Processing Systems 12, S. A. Solla, T. K. Leen, K.-R. Mller, eds*, Cambridge, MA, 2000. MIT Press.

[9] V. N. Vapnik. *Statistical Learning Theory*. John Wiley & Sons, New-York, 1998.
